# ICA with Reconstruction Cost for Efficient Overcomplete Feature Learning

**Quoc V. Le, Alexandre Karpenko, Jiquan Ngiam and Andrew Y. Ng**
*{quocle,akarpenko,jngiam,ang}@cs.stanford.edu*
Computer Science Department, Stanford University

## Abstract

Independent Components Analysis (ICA) and its variants have been successfully used for unsupervised feature learning. However, standard ICA requires an orthonoramlity constraint to be enforced, which makes it difficult to learn overcomplete features. In addition, ICA is sensitive to whitening. These properties make it challenging to scale ICA to high dimensional data. In this paper, we propose a robust soft reconstruction cost for ICA that allows us to learn highly overcomplete sparse features even on unwhitened data. Our formulation reveals formal connections between ICA and sparse autoencoders, which have previously been observed only empirically. Our algorithm can be used in conjunction with off-the-shelf fast unconstrained optimizers. We show that the soft reconstruction cost can also be used to prevent replicated features in tiled convolutional neural networks. Using our method to learn highly overcomplete sparse features and tiled convolutional neural networks, we obtain competitive performances on a wide variety of object recognition tasks. We achieve state-of-the-art test accuracies on the STL-10 and Hollywood2 datasets.

## 1 Introduction

Sparsity has been shown to work well for learning feature representations that are robust for object recognition [1, 2, 3, 4, 5, 6, 7]. A number of algorithms have been proposed to learn sparse features. These include: sparse auto-encoders [8], Restricted Boltzmann Machines (RBMs) [9], sparse coding [10] and Independent Component Analysis (ICA) [11]. ICA, in particular, has been shown to perform well in a wide range of object recognition tasks [12]. In addition, ISA (Independent Subspace Analysis, a variant of ICA) has been used to learn features that achieved state-of-the-art performance on action recognition tasks [13].

However, standard ICA has two major drawbacks. First, it is difficult to learn *overcomplete feature representations* (i.e., the number of features cannot exceed the dimensionality of the input data). This puts ICA at a disadvantage compared to other methods, because Coates et al. [6] have shown that classification performance improves for algorithms such as sparse autoencoders [8], K-means [6] and RBMs [9], when the learned features are overcomplete. Second, ICA is sensitive to *whitening* (a preprocessing step that decorrelates the input data, and cannot always be computed exactly for high dimensional data). As a result, it is difficult to scale ICA to high dimensional data. In this paper we propose a modification to ICA that not only addresses these shortcomings but also reveals strong connections between ICA, sparse autoencoders and sparse coding.

Both drawbacks arise from a constraint in the standard ICA formulation that requires features to be orthogonal. This hard orthonormality constraint, $WW^T = \mathbf{I}$, is used to prevent degenerate solutions in the feature matrix $W$ (where each feature is a row of $W$). However, if $W$ is overcomplete (i.e., a "tall" matrix) then this constraint can no longer be satisfied. In particular, the standard optimization procedure for ICA, ISA and TICA (Topographic ICA) uses projected gradient descent, where W is

orthonormalized at each iteration by solving $W := (WW^T)^{-\frac{1}{2}}W$. This symmetric orthonormalization procedure does not work when $W$ is overcomplete. As a result, this standard ICA method can not learn more features than the number of dimensions in the data. Furthermore, while alternative orthonormalization procedures or score matching can learn overcomplete representations, they are expensive to compute. Constrained optimizers also tend to be much slower than unconstrained ones.[1]

Our algorithm enables ICA to scale to overcomplete representations by replacing the orthonormalization constraint with a linear reconstruction penalty (akin to the one used in sparse auto-encoders). This reconstruction penalty removes the need for a constrained optimizer. As a result, we can implement our algorithm with only a few lines of MATLAB, and plug it directly into unconstrained solvers (e.g., L-BFGS and CG [14]). This results in very fast convergence rates for our method.

In addition, recent ICA-based algorithms, such as tiled convolutional neural networks (also known as local receptive field TICA) [12], also suffer from the difficulty of enforcing the hard orthonormality constraint globally. As a result, orthonormalization is typically performed locally instead, which results in copied (i.e., degenerate) features. Our reconstruction penalty, on the other hand, can be enforced globally across all receptive fields. As a result, our method prevents degenerate features.

Furthermore, ICA's sensitivity to whitening is undesirable because exactly whitening high dimensional data is often not feasible. For example, exact whitening using principal component analysis (PCA) for input images of size 200x200 pixels is challenging, because it requires solving the eigendecomposition of a 40,000 x 40,000 covariance matrix. Other methods, such as sparse autoencoders or RBMs, work well using approximate whitening and in some cases work even without any whitening. Standard ICA, on the other hand, tends to produce noisy filters unless the data is exactly white. Our soft-reconstruction penalty shares the property of auto-encoders, in that it makes our approach also less sensitive to whitening. Similarities between ICA, auto-encoders and sparse coding have been observed empirically before (i.e., they all learn edge filters). Our contribution is to show a formal proof and a set of conditions under which these algorithms are equivalent.

Finally, we use our algorithm for classifying STL-10 images [6] and Hollywood2 [15] videos. In particular, on the STL-10 dataset, we learn highly overcomplete representations and achieve 52.9% on the test set. On Hollywood2, we achieve 54.6 Mean Average Precision, which is also the best published result on this dataset.

## 2 Standard ICA and Reconstruction ICA

We begin by introducing our proposed algorithm for overcomplete ICA. In subsequent sections we will show how our method is related to ICA, sparse auto-encoders and sparse coding. Given unlabeled data $\{x^{(i)}\}_{i=1}^m, x^{(i)} \in \mathbb{R}^n$, regular ICA [11] is traditionally defined as the following optimization problem:

$$\underset{W}{\text{minimize}} \ \sum_{i=1}^m \sum_{j=1}^k g(W_j x^{(i)}), \ \text{subject to} \ WW^T = \mathbf{I} \tag{1}$$

where $g$ is a nonlinear convex function, e.g., smooth $L_1$ penalty: $g(.) := \log(\cosh(.))$ [16], $W$ is the weight matrix $W \in \mathbb{R}^{k \times n}$, $k$ is number of components (features), and $W_j$ is one row (feature) in $W$. The orthonormality constraint $WW^T = \mathbf{I}$ is used to prevent the bases in $W$ from becoming degenerate. We refer to this as "non-degeneracy control" in this paper.

Typically, ICA requires data to have zero mean, $\sum_{i=1}^m x^{(i)} = 0$, and unit covariance, $\frac{1}{m}\sum_{i=1}^m x^{(i)}(x^{(i)})^T = \mathbf{I}$. While the former can be achieved by subtracting the empirical mean, the latter requires finding a linear transformation by solving the eigendecomposition of the covariance matrix [11]. This preprocessing step is also known as whitening or sphering the data.

For overcomplete representations ($k > n$) [17, 18], the orthonormality constraint can no longer hold. As a result, approximate orthonormalization (e.g., Gram-Schmidt) or fixed-point iterative

methods [11] have been proposed. These algorithms are often slow and require tuning. Other approaches, e.g., interior point methods [19] or score matching [16] exist, but they are complicated to implement and also slow. Score matching, for example, is difficult to implement and expensive for multilayered algorithms like ISA or TICA, because it requires backpropagation of a Hessian matrix.

These challenges motivate our search for a better type of non-degeneracy control for ICA. A frequently employed form of non-degeneracy control in auto-encoders and sparse coding is the use of reconstruction costs. As a result, we propose to replace the hard orthonormal constraint in ICA with a soft reconstruction cost. Applying this change to eq. 1, produces the following unconstrained problem:

$$\textbf{Reconstruction ICA (RICA):} \quad \underset{W}{\text{minimize}} \; \frac{\lambda}{m} \sum_{i=1}^{m} \|W^T W x^{(i)} - x^{(i)}\|_2^2 + \sum_{i=1}^{m} \sum_{j=1}^{k} g(W_j x^{(i)}) \qquad (2)$$

We use the term "reconstruction cost" for this smooth penalty because it corresponds to the reconstruction cost of a linear autoencoder, where the encoding weights and decoding weights are tied (i.e., the encoding step is $W x^{(i)}$ and the decoding step is $W^T W x^{(i)}$).

The choice to swap the orthonormality constraint with a reconstruction penalty seems arbitrary at first. However, we will show in the following section that these two forms of degeneracy control are, in fact, equivalent under certain conditions. Furthermore, this change has two key benefits: first, it allows unconstrained optimizers (e.g., L-BFGS, CG [20] and SGDs) to be used to minimize this cost function instead of relying on slower constrained optimizers (e.g., projected gradient descent) to solve the standard ICA cost function. And second, the reconstruction penalty works even when W is overcomplete and the data not fully white.

## 3   Connections between orthonormality and reconstruction

Sparse autoencoders, sparse coding and ICA have been previously suspected to be strongly connected because they learn edge filters for natural image data. In this section we present formal proofs that they are indeed mathematically equivalent under certain conditions (e.g., whitening and linear coding). Our proofs reveal the underlying principles in unsupervised feature learning that tie these algorithms together.

We start by reviewing the optimization problems of two common unsupervised feature learning algorithms: sparse autoencoders and sparse coding. In particular, the objective function of tied-weight sparse autoencoders [8, 21, 22, 23] is:

$$\underset{W,b,c}{\text{minimize}} \; \frac{\lambda}{m} \sum_{i=1}^{m} \|\sigma(W^T \sigma(W x^{(i)} + b) + c) - x^{(i)}\|_2^2 + S(\{W,b\}, x^{(1)}, \dots, x^{(m)}) \qquad (3)$$

where $\sigma$ is the activation function (e.g., sigmoid), $b, c$ are biases, and $S$ is some sparse penalty function. Typically, $S$ is chosen to be the smooth $L_1$ penalty $S(\{W,b\}, x^{(i)}, \dots, x^{(m)}) = \sum_{i=1}^{m} \sum_{j=1}^{k} g(W_j x^{(i)})$ or KL divergence between the average activation and target activation [24].

Similarly, the optimization problem of Sparse coding [10] is:

$$\underset{W, z^{(1)}, \dots, z^{(m)}}{\text{minimize}} \; \frac{\lambda}{m} \sum_{i=1}^{m} \|W^T z^{(i)} - x^{(i)}\|_2^2 + \sum_{i=1}^{m} \sum_{j=1}^{k} g(z_j^{(i)}) \;\; \text{subject to} \quad \|W_j\|_2^2 \le c, \forall j = 1, \dots, k. \quad (4)$$

From these formulations, it is clear there are links between ICA, RICA, sparse autoencoders and sparse coding. In particular, most methods use the $L_1$ sparsity penalty and, except for ICA, most use reconstruction costs as a non-degeneracy control. These observations are summarized in Table 1.

ICA's main distinction compared to sparse coding and autoencoders is its use of the hard orthonormality constraint in lieu of reconstruction costs. However, we will now present a proof (consisting of two lemmas) that derives the relationship between ICA's orthonormality constraint and RICA's reconstruction cost. We subsequently present a set of conditions under which RICA is equivalent to sparse coding and autoencoders. The result is a novel and formal proof of the relationship between ICA, sparse coding and autoencoders.

We let $\mathbf{I}$ denote an identity matrix, and $\mathbf{I}_l$ an identity matrix of size $l \times l$. We denote the $L_2$ norm by $\|.\|_2$ and the matrix Frobenius norm by $\|.\|_{\mathcal{F}}$. We also assume that the data $\{x^{(i)}\}_{i=1}^{m}$ has zero mean.

Table 1: A summary of different unsupervised feature learning methods. "Non-degeneracy control" refers to the mechanism that prevents all bases from learning uninteresting weights (e.g., zero weights or identical weights). Note that using sparsity is optional in autoencoders.

| Algorithm | Sparsity | Non-degeneracy control | Activation function |
|---|---|---|---|
| Sparse coding [10] | $L_1$ | $L_2$ reconstruction | Implicit |
| Autoencoders and Denoising autoencoders [21] | Optional: KL [24] or $L_1$ [22] | $L_2$ reconstruction (or cross entropy [21, 8]) | Sigmoid |
| ICA [16] | $L_1$ | Orthonormality | Linear |
| RICA (this paper) | $L_1$ | $L_2$ reconstruction | Linear |

The first lemma states that the reconstruction cost and column orthonormality cost[2] are equivalent when data is whitened (see the Appendix in the supplementary material for proofs):

**Lemma 3.1** *When the input data $\{x^{(i)}\}_{i=1}^m$ is whitened, the reconstruction cost $\frac{\lambda}{m}\sum_{i=1}^m \|W^T W x^{(i)} - x^{(i)}\|_2^2$ is equivalent to the orthonormality cost $\lambda\|W^T W - \mathbf{I}\|_{\mathcal{F}}^2$.*

Our second lemma states that minimizing column orthonormality and row orthonormality costs turns out to be equivalent due to a property of the Frobenius norm:

**Lemma 3.2** *The column orthonormality cost $\lambda\|W^T W - \mathbf{I}_n\|_{\mathcal{F}}^2$ is equivalent to the row orthonormality cost $\lambda\|WW^T - \mathbf{I}_k\|_{\mathcal{F}}^2$ up to an additive constant.*

Together these two lemmas tell us that reconstruction cost is equivalent to both column and row orthonormality cost for whitened data. Furthermore, as $\lambda$ approaches infinity the orthonormality cost becomes the hard orthonormality constraint of ICA (see equations 1 & 2) if $W$ is complete or undercomplete. Thus, ICA's hard orthonormality constraint and RICA's reconstruction cost are related under these conditions. More formally, the following remarks explain this conclusion, and describe the set of conditions under which RICA (and by extension ICA) is equivalent to autoencoders and sparse coding.

1) If the data is whitened, RICA is equivalent to ICA for undercomplete representations and $\lambda$ approaching infinity. For whitened data our RICA formulation:

$$\textbf{RICA:}\quad \underset{W}{\text{minimize}}\ \frac{\lambda}{m}\sum_{i=1}^m \|W^T W x^{(i)} - x^{(i)}\|_2^2 + \sum_{i=1}^m \sum_{j=1}^k g(W_j x^{(i)}) \tag{5}$$

is equivalent (from the above lemmas) to:

$$\underset{W}{\text{minimize}}\ \lambda\|W^T W - \mathbf{I}\|_{\mathcal{F}}^2 + \sum_{i=1}^m \sum_{j=1}^k g(W_j x^{(i)}),\ \text{and} \tag{6}$$

$$\underset{W}{\text{minimize}}\ \lambda\|WW^T - \mathbf{I}\|_{\mathcal{F}}^2 + \sum_{i=1}^m \sum_{j=1}^k g(W_j x^{(i)}) \tag{7}$$

Furthermore, for undercomplete representations, in the limit of $\lambda$ approaching infinity, the orthonormalization costs above become hard constraints. As a result, they are equivalent to:

$$\textbf{Conventional ICA:}\quad \sum_{i=1}^m \sum_{j=1}^k g(W_j x^{(i)})\ \text{subject to}\ WW^T = \mathbf{I} \tag{8}$$

which is just plain ICA, or ISA/TICA with appropriate choices of the sparsity function $g$.

2) Autoencoders and Sparse Coding are equivalent to RICA if

- in autoencoders, we use a linear activation function $\sigma(x) = x$, ignore the biases $b, c$, use the soft $L_1$ sparsity for the activations: $S(\{W,b\}, x^{(i)}, \ldots, x^{(m)}) = \sum_{i=1}^m \sum_{j=1}^k g(W_j x^{(i)})$ and

- in sparse coding, we use explicit encoding $z_j^{(i)} = W_j x^{(i)}$ and ignore the norm ball constraints.

Despite their equivalence, certain formulations have certain advantages. For instance, RICA (eq. 2) and soft orthonormalization ICA (eq. 6 and 7) are smooth and can be optimized efficiently by fast unconstrained solvers (e.g., L-BFGS or CG) while the conventional constrained ICA optimization problem cannot. Soft penalties are also preferred if we want to learn overcomplete representations where explicitly constraining $WW^T = \mathbf{I}$ is not possible[3].

We derive an additional relationship in the appendix (see supplementary material), which shows that for whitened data denoising autoencoders are equivalent to RICA with weight decay. Another interesting connection between RBMs and denoising autoencoders is derived in [25]. The connections, between RBMs, autoencoders, denoising autoencoders and the fact that reconstruction cost captures whitening (by the above lemmas), likely explains why whitening does not matter much for RBMs and autoencoders in [6].

## 4 Effects of whitening on ICA and RICA

In practice, ICA tends to be much more sensitive to whitening compared to sparse autoencoders. Running ICA on unwhitened data results in very noisy bases. In this section, we study empirically how whitening affects ICA and our formulation, RICA.

We sampled 20000 patches of size 16x16 from a set of 11 natural images [16] and visualized the filters learned using ICA and RICA with raw images, as well as approximately whitened images. For approximate whitening, we use 1/f whitening with low pass filtering. This 1/f whitening transformation uses Fourier analysis of natural image statistics and produces transformed data which has an approximate identity covariance matrix. This procedure does not require pretraining. As a result, 1/f whitening runs quickly and scales well to high dimensional data. We used the 1/f whitening implementation described in [16].

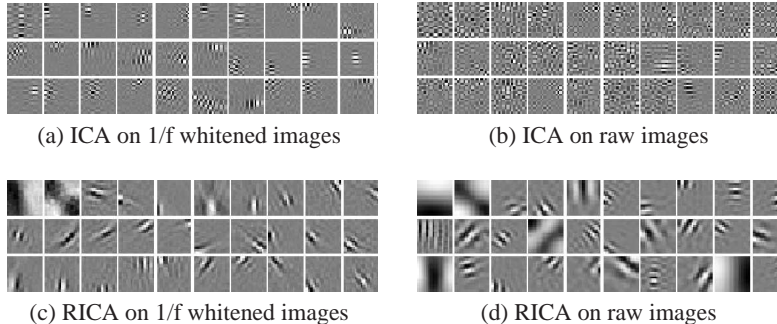

(a) ICA on 1/f whitened images          (b) ICA on raw images

(c) RICA on 1/f whitened images        (d) RICA on raw images

Figure 1: ICA and RICA on approximately whitened and raw images. (a-b): Bases learned with ICA. (c-d): Bases learned with RICA. RICA retains some structures of the data whereas ICA does not (i.e., it learns noisy bases).

Figure 1 shows the results of running ICA and RICA on raw and 1/f whitened images. As can be seen, ICA learns very noisy bases on raw data, as well as approximately whitened data. In contrast, RICA works well for 1/f whitened data and raw data. Our quantitative analysis with kurtosis (not shown due to space limits) agrees with visual inspection: RICA learns more kurtotic representations than ICA on approximately whitened or raw data.

Robustness to approximate whitening is desirable, because exactly whitening high dimensional data using PCA may not be feasible. For instance, PCA on images of size 200x200 requires computing the eigendecomposition of a 40,000 x 40,000 covariance matrix, which is computationally expensive. With RICA, approximate whitening or raw data can be used instead. This allows our method to scale to higher dimensional data than regular ICA.

## 5 Local receptive field TICA

The first application of our RICA algorithm that we examine is local receptive field neural networks. The motivation behind local receptive fields is computational efficiency. Specifically, rather

than having each hidden unit connect to the entire input image, each unit is instead connected to a small patch (see figure 2a for an illustration). This reduces the number of parameters in the model. As a result, local receptive field neural networks are faster to optimize than their fully connected counterparts. A major drawback of this approach, however, is the difficulty in enforcing orthogonality across partially overlapping patches. We show that swapping out locally enforced orthogonality constraints with a global reconstruction cost solves this issue.

Specifically, we examine the local receptive field network proposed by Le et al. [12]. Their formulation constrains each feature (a row of $W$) to connect to a small region of the image (i.e., all weights outside of the patch are set to zero). This modification allows learning ICA and TICA with larger images, because $W$ is now sparse. Unlike standard convolutional networks, these networks may be extended to have fully unshared weights. This permits them to learn invariances other than translational invariances, which are hardwired in convolutional networks.

The pre-training step for the TCNN (local receptive field TICA) [12] is performed by minimizing the following cost function:

$$\underset{W}{\text{minimize}} \quad \sum_{i=1}^{m}\sum_{j=1}^{k}\sqrt{\epsilon + H_j(Wx^{(i)})^2}, \text{ subject to } WW^T = \mathbf{I} \tag{9}$$

where $H$ is the spatial pooling matrix and $W$ is a learned weight matrix. The corresponding neural network representation of this algorithm is one with two layers with weights $W, H$ and nonlinearities $(.)^2$ and $\sqrt{(.)}$ respectively (see Figure 2a). In addition, $W$ and $H$ are set to be local. That is, each row of $W$ and $H$ connects to a small region of the input data.

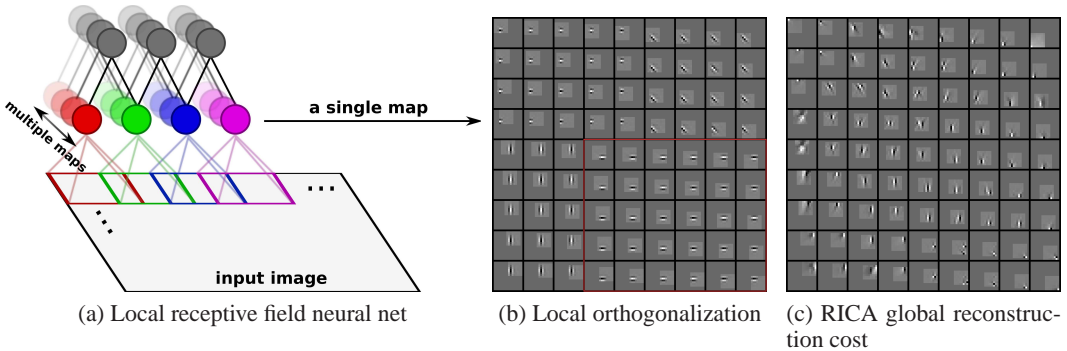

(a) Local receptive field neural net     (b) Local orthogonalization     (c) RICA global reconstruction cost

Figure 2: (a) Local receptive field neural network with fully untied weights. A single map consists of local receptive fields that do not share a location (i.e., only different colored nodes). (b & c) For illustration purposes we have brightened the area of each local receptive field within the input image. (b) Hard orthonormalization [12] is applied at each location only (i.e., nodes of the same color), which results in copied filters (for example, see the filters outlined in red; notice that the location of the edge stays the same within the image even though the receptive field areas are different). (c) Global reconstruction (this paper) is applied both within each location and across locations (nodes of the same and different colors), which prevents copying of receptive fields.

Enforcing the hard orthonormality constraint on the entire sparse $W$ matrix is challenging because it is typically overcomplete for TCNNs. As a result, Le et al. [12] performed local orthonormalization instead. That is, only the features (rows of $W$) that share a location (e.g., only the red nodes in figure 2) were orthonormalized using symmetric orthogonalization.

However, visualizing the filters learned by a TCNN with local orthonormalization, shows that many adjacent receptive fields end up learning the same (copied) filters due to the lack of an orthonormality constraint between them. For instance, the green nodes in Figure 2 may end up being copies of the red nodes (see the copied receptive fields in Figure 2b).

In order to prevent copied features, we replace the local orthonormalization constraint with a global reconstruction cost (i.e., computing the reconstruction cost $\|W^TWx^{(i)} - x^{(i)}\|_2^2$ for the entire overcomplete sparse $W$ matrix). Figure 2c shows the resulting filters. Figure 3 shows that the reconstruction penalty produces a better distribution of edge detector locations within the image patch (this also holds true for frequencies and orientations).

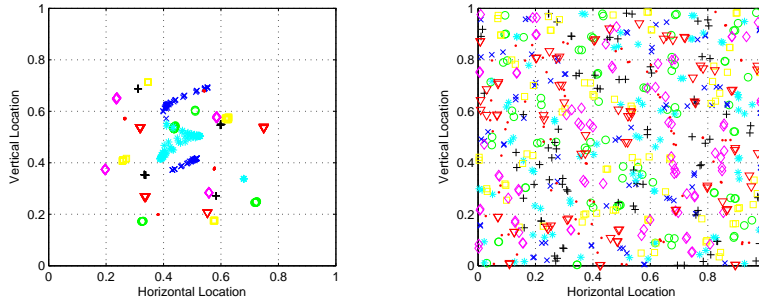

Figure 3: Location of each edge detector within the image patch. Symbols of the same color/shape correspond to a single map. *Left:* local orthonormalization constraint. *Right:* global reconstruction penalty. The reconstruction penalty prevents copied filters, producing a more uniform distribution of edge detectors.

# 6 Experiments

The following experiments compare the speed gains of RICA over standard overcomplete ICA. We then use RICA to learn a large filter bank, and show that it works well for classification on the STL-10 dataset.

## 6.1 Speed improvements for overcomplete ICA

In this experiment, we examine the speed performance of RICA and overcomplete ICA with score matching [26]. We trained overcomplete ICA on 20000 gray-scale image patches, each patch of size 16x16. We learn representations that are 2x, 4x and 6x overcomplete. We terminate both algorithms when changes in the parameter vector drop below $10^{-6}$. We use the score matching implementation provided in [16]. We report the time required to learn these representations in Table 2. The results show that our method is much faster than the competing method. In particular, learning features that are 6x overcomplete takes 1 hour using our method, whereas [26] requires 2 days.

Table 2: Speed improvements of our method over score matching [26].

|  | 2x overcomplete | 4x overcomplete | 6x overcomplete |
|---|---|---|---|
| Score matching ICA | 33000 seconds | 65000 seconds | 180000 seconds |
| RICA | 1000 seconds | 1600 seconds | 3700 seconds |
| Speed up | 33x | 40x | 48x |

Figure 4 shows the peak frequencies and orientations for 4x overcomplete bases learned using our method. The learned bases do not degenerate, and they cover a broad range of frequencies and orientations (cf. Figure 3 in [27]). This ability to learn a diverse set of features allows our algorithm to perform well on various discriminative tasks.

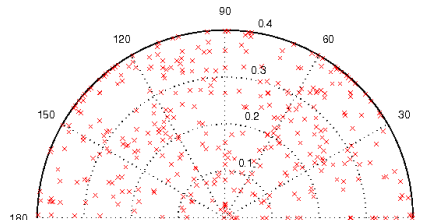

Figure 4: Scatter plot of peak frequencies and orientations of Gabor functions fitted to the filters learned by RICA on whitened images. Our model yields a diverse set of filters that covers the spatial frequency space evenly.

## 6.2 Overcomplete ICA on STL-10 dataset

In this section, we evaluate the overcomplete features learned by our model. The experiments are carried out on the STL-10 dataset [6] where overcomplete representations have been shown to work well. The STL-10 dataset contains 96x96 pixel color images taken from 10 classes. For each

class 500 training images and 800 test images are provided. In addition, 100,000 unlabeled images are included for unsupervised learning. We use RICA to learn overcomplete features on 100,000 randomly sampled color patches from the unlabeled images in the STL-10 dataset. We then apply RICA to extract features from images in the same manner described in [6].

Using the same number of features (1600) employed by Coates et al. [6] on 96x96 images and 10x10 receptive fields, our soft reconstruction ICA achieves 52.9% on the test set. This result is slightly better than (but within the error bars) of the best published result, 51.5%, obtained by K-means [6].

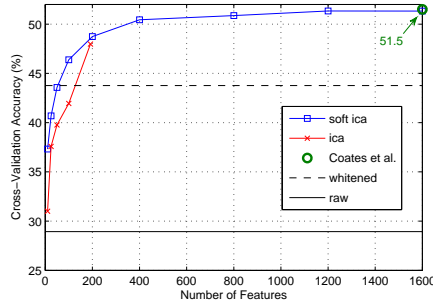

Figure 5: Classification accuracy on the STL-10 dataset as a function of the number of bases learned (for a patch size of 8x8 pixels). The best result shown uses bases that are 8x overcomplete.

Finally, we compare classification accuracy as a function of the number of bases. Figure 5 shows the results for ICA and RICA. Notice that the reconstruction cost in RICA allows us to learn overcomplete representations that outperform the complete representation obtained by the regular ICA.

### 6.3 Reconstruction Independent Subspace Analysis for action recognition

Recently we presented a system [13] for learning features from unlabelled data that can lead to state-of-the-art performance on many challenging datasets such as Hollywood2 [15], KTH [28] and YouTube [29]. This system makes use of a two-layered Independent Subspace Analysis (ISA) network [16]. Like ICA, ISA also uses orthogonalization for degeneracy control.[4]

In this section we compare the effects of reconstruction versus orthogonality on classification performance using ISA. In our experiments we swap out the orthonormality constraint employed by ISA with a reconstruction penalty. Apart from this change, the entire pipeline and parameters are identical to the system described in [13].

We observe that the reconstruction penalty tends to works better than orthogonality constraints. In particular, on the Hollywood2 dataset ISA achieves a mean AP of 54.6% when the reconstruction penalty is used. The performance of ISA drops to 53.3% when orthogonality constraints are used. Both results are state-of-the art resuls on this dataset [30]. We attribute the improvement in performance to the fact that features in invariant subspaces of ISA need not be strictly orthogonal.

## 7    Discussion

In this paper, we presented a novel soft reconstruction approach that enables the learning of overcomplete representations in ICA and TICA. We have also presented mathematical proofs that connect ICA with autoencoders and sparse coding. We showed that our algorithm works well even without whitening; and that the reconstruction cost allows us to fix replicated filters in tiled convolutional neural networks. Our experiments show that RICA is fast and works well in practice. In particular, we found our method to be 30-50x faster than overcomplete ICA with score matching. Furthermore, our overcomplete features achieve state-of-the-art performance on the STL-10 and Hollywood2 datasets.

## Footnotes

[1]FastICA is a specialized solver that works well for complete or undercomplete ICA. Here, we focus our attention on ICA and its variants such as ISA and TICA in the context of overcomplete representations, where FastICA does not work.

[2]The column orthonormality cost is zero only if the columns of $W$ are orthonormal.

[3]Note that when $W$ is overcomplete, some rows may degenerate and become zero, because the reconstruction constraint can be satisfied with only a complete subset of rows. To prevent this, we employ an additional norm ball constraint (see the Appendix for more details regarding L-BFGS and norm ball constraints).

[4]Note that in ISA the square nonlinearity is used in the first layer, and squareroot is used in in second layer [13].

# References

[1] M.A. Ranzato, C. Poultney, S. Chopra, and Y. LeCun. Efficient learning of sparse representations with an energy-based model. In *NIPS*, 2006.

[2] R. Raina, A. Battle, H. Lee, B. Packer, and A.Y. Ng. Self-taught learning: Transfer learning from unlabelled data. In *ICML*, 2007.

[3] M. Ranzato, F. J. Huang, Y. Boureau, and Y. LeCun. Unsupervised learning of invariant feature hierarchies with applications to object recognition. In *CVPR*, 2007.

[4] J. Yang, K. Yu, Y. Gong, and T. Huang. Linear spatial pyramid matching using sparse coding for image classification. In *CVPR*, 2009.

[5] J. Yang, K. Yu, and T. Huang. Efficient highly over-complete sparse coding using a mixture model. In *ECCV*, 2010.

[6] A. Coates, H. Lee, and A. Y. Ng. An analysis of single-layer networks in unsupervised feature learning. In *AISTATS 14*, 2011.

[7] K. Yu, Y. Lin, and J. Lafferty. Learning image representations from pixel level via hierarchical sparse coding. In *CVPR*, 2011.

[8] Y. Bengio, P. Lamblin, D. Popovici, and H. Larochelle. Greedy layerwise training of deep networks. In *NIPS*, 2007.

[9] G. E. Hinton, S. Osindero, and Y. W. Teh. A fast learning algorithm for deep belief nets. *Neural Computation*, 2006.

[10] B. Olshausen and D. Field. Emergence of simple-cell receptive field properties by learning a sparse code for natural images. *Nature*, 1996.

[11] A. Hyvärinen, J. Karhunen, and E. Oja. *Independent Component Analysis*. Wiley Interscience, 2001.

[12] Q. V. Le, J. Ngiam, Z. Chen, D. Chia, P. W. Koh, and A. Y. Ng. Tiled convolutional neural networks. In *NIPS*, 2010.

[13] Q. V. Le, W. Zou, S. Y. Yeung, and A. Y. Ng. Learning hierarchical spatio-temporal features for action recognition with independent subspace analysis. In *CVPR*, 2011.

[14] Q. V. Le, J. Ngiam, A. Coates, A. Lahiri, B. Prochnow, and A. Y. Ng. On optimization methods for deep learning. In *ICML*, 2011.

[15] M. Marzalek, I. Laptev, and C. Schmid. Actions in context. In *CVPR*, 2009.

[16] A. Hyvärinen, J. Hurri, and P. O. Hoyer. *Natural Image Statistics*. Springer, 2009.

[17] B. Olshausen and D. Field. Sparse coding with an overcomplete basis set: A strategy employed by v1. *Vision Research*, 1997.

[18] M. S. Lewicki and T. J. Sejnowski. Learning overcomplete representations. *Neural Computation*, 2000.

[19] L. Ma and L. Zhang. Overcomplete topographic independent component analysis. *Elsevier*, 2008.

[20] M. Schmidt. minFunc, 2005.

[21] P. Vincent, H. Larochelle, Y. Bengio, and P. A. Manzagol. Extracting and composing robust features with denoising autoencoders. In *ICML*, 2008.

[22] H. Lee, C. Ekanadham, and A. Y. Ng. Sparse deep belief net model for visual area V2. In *NIPS*, 2008.

[23] H. Larochelle, Y. Bengio, J. Louradour, and P. Lamblin. Exploring strategies for training deep neural networks. *JMLR*, 2009.

[24] G. Hinton. A practical guide to training restricted boltzmann machines. Technical report, U. of Toronto, 2010.

[25] P. Vincent. A connection between score matching and denoising autoencoders. *Neural Computation*, 2010.

[26] A. Hyvärinen. Estimation of non-normalized statistical models using score matching. *JMLR*, 2005.

[27] Y. Karklin and M.S. Lewicki. Is early vision optimized for extracting higher-order dependencies? In *NIPS*, 2006.

[28] C. Schuldt, I. Laptev, and B. Caputo. Recognizing human actions: A local SVM approach. In *ICPR*, 2004.

[29] J. Liu, J. Luo, and M. Shah. Recognizing realistic actions from videos "in the Wild". In *CVPR*, 2009.

[30] Heng Wang, Muhammad Muneeb Ullah, Alexander Klaser, Ivan Laptev, and Cordelia Schmid. Evaluation of local spatio-temporal features for action recognition. In *BMVC*, 2010.

